# Two Iterative Algorithms for Computing the Singular Value Decomposition from Input/Output Samples

**Terence D. Sanger**
Jet Propulsion Laboratory
MS 303-310
4800 Oak Grove Drive
Pasadena, CA 91109

## Abstract

The Singular Value Decomposition (SVD) is an important tool for linear algebra and can be used to invert or approximate matrices. Although many authors use "SVD" synonymously with "Eigenvector Decomposition" or "Principal Components Transform", it is important to realize that these other methods apply only to symmetric matrices, while the SVD can be applied to arbitrary nonsquare matrices. This property is important for applications to signal transmission and control.

I propose two new algorithms for iterative computation of the SVD given only sample inputs and outputs from a matrix. Although there currently exist many algorithms for Eigenvector Decomposition (Sanger 1989, for example), these are the first true sample-based SVD algorithms.

## 1 INTRODUCTION

The Singular Value Decomposition (SVD) is a method for writing an arbitrary nonsquare matrix as the product of two orthogonal matrices and a diagonal matrix. This technique is an important component of methods for approximating near-singular matrices and computing pseudo-inverses. Several efficient techniques exist for finding the SVD of a known matrix (Golub and Van Loan 1983, for example).

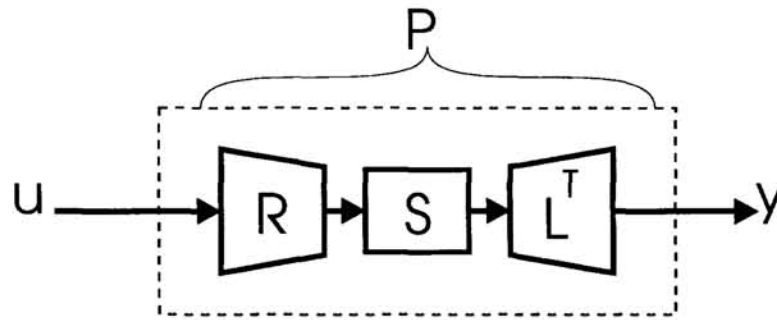

Figure 1: Representation of the plant matrix $P$ as a linear system mapping inputs $u$ into outputs $y$. $L^T S R$ is the singular value decomposition of $P$.

However, for certain signal processing or control tasks, we might wish to find the SVD of an *unknown* matrix for which only input-output samples are available. For example, if we want to model a linear transmission channel with unknown properties, it would be useful to be able to approximate the SVD based on samples of the inputs and outputs of the channel. If the channel is time-varying, an iterative algorithm for approximating the SVD might be able to track slow variations.

## 2   THE SINGULAR VALUE DECOMPOSITION

The SVD of a nonsymmetric matrix $P$ is given by $P = L^T S R$ where $L$ and $R$ are matrices with orthogonal rows containing the left and right "singular vectors", and $S$ is a diagonal matrix of "singular values". The inverse of $P$ can be computed by inverting $S$, and approximations to $P$ can be formed by setting the values of the smallest elements of $S$ to zero.

For a memoryless linear system with inputs $u$ and outputs $y = Pu$, we can write $y = L^T S R u$ which shows that $R$ gives the "input" transformation from inputs to internal "modes", $S$ gives the gain of the modes, and $L^T$ gives the "output" transformation which determines the effect of each mode on the output. Figure 1 shows a representation of this arrangement.

The goal of the two algorithms presented below is to train two linear neural networks $N$ and $G$ to find the SVD of $P$. In particular, the networks attempt to invert $P$ by finding orthogonal matrices $N$ and $G$ such that $NG \approx P^{-1}$, or $PNG = I$. A particular advantage of using the iterative algorithms described below is that it is possible to extract only the singular vectors associated with the largest singular values. Figure 2 depicts this situation, in which the matrix $S$ is shown smaller to indicate a small number of significant singular values.

There is a close relationship with algorithms that find the eigenvalues of a symmetric matrix, since any such algorithm can be applied to $PP^T = L^T S^2 L$ and $P^T P = R^T S^2 R$ in order to find the left and right singular vectors. But in a behaving animal or operating robotic system it is generally not possible to compute the product with $P^T$, since the plant is an unknown component of the system. In the following, I will present two new iterative algorithms for finding the singular value decomposition

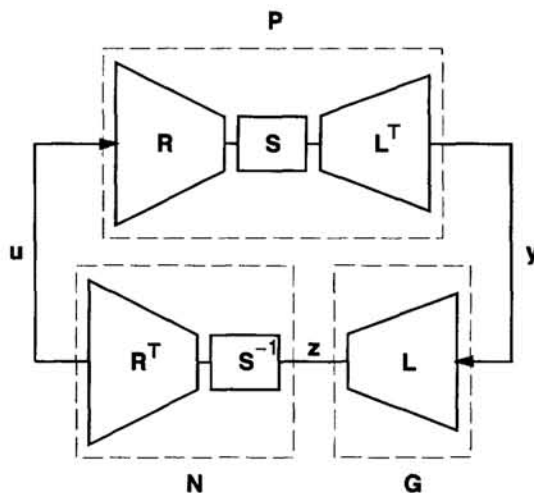

Figure 2: Loop structure of the singular value decomposition for control. The plant is $P = L^T SR$, where $R$ determines the mapping from control variables to system modes, and $L^T$ determines the outputs produced by each mode. The optimal sensory network is $G = L$, and the optimal motor network is $N = R^T S^{-1}$. $R$ and $L$ are shown as trapezoids to indicate that the number of nonzero elements of $S$ (the "modes") may be less than the number of sensory variables $y$ or motor variables $u$.

of a matrix $P$ given only samples of the inputs $u$ and outputs $y$.

## 3   THE DOUBLE GENERALIZED HEBBIAN ALGORITHM

The first algorithm is the Double Generalized Hebbian Algorithm (DGHA), and it is described by the two coupled difference equations

$$\Delta G = \gamma(zy^T - \mathrm{LT}[zz^T]G) \tag{1}$$

$$\Delta N^T = \gamma(zu^T - \mathrm{LT}[zz^T]N^T) \tag{2}$$

where LT[ ] is an operator that sets the above diagonal elements of its matrix argument to zero, $y = Pu$, $z = Gy$, and $\gamma$ is a learning rate constant.

Equation 1 is the Generalized Hebbian Algorithm (Sanger 1989) which finds the eigenvectors of the autocorrelation matrix of its inputs $y$. For random uncorrelated inputs $u$, the autocorrelation of $y$ is $E[yy^T] = L^T S^2 L$, so equation 1 will cause $G$ to converge to the matrix of left singular vectors $L$. Equation 2 is related to the Widrow-Hoff (1960) LMS rule for approximating $u^T$ from $z$, but it enforces orthogonality of the columns of $N$. It appears similar in form to equation 1, except that the intermediate variables $z$ are computed from $y$ rather than $u$. A graphical representation of the algorithm is given in figure 3. Equations 1 and 2 together cause $N$ to converge to $R^T S^{-1}$, so that the combination $NG = R^T S^{-1} L$ is an approximation to the plant inverse.

**Theorem 1:** *(Sanger 1993) If $y = Pu$, $z = Gy$, and $E[uu^T] = I$, then equations 1 and 2 converge to the left and right singular vectors of $P$.*

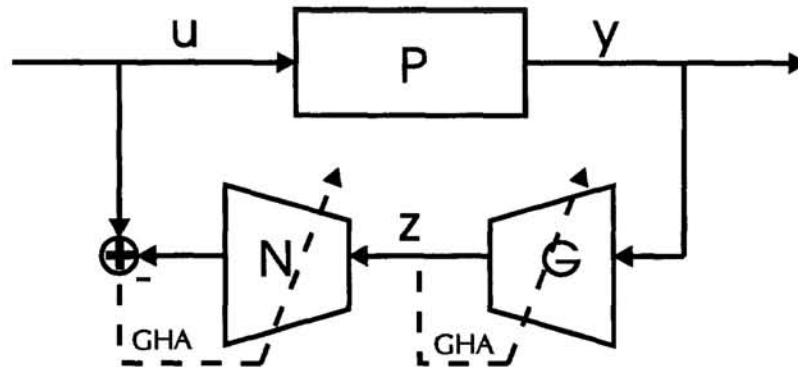

Figure 3: Graphic representation of the Double Generalized Hebbian Algorithm. $G$ learns according to the usual GHA rule, while $N$ learns using an orthogonalized form of the Widrow-Hoff LMS Rule.

Proof:

After convergence of equation 1, $E[zz^T]$ will be diagonal, so that $E[LT[zz^T]] = E[zz^T]$. Consider the Widrow-Hoff LMS rule for approximating $u^T$ from $z$:

$$\Delta N^T = \gamma(zu^T - zz^T N^T).\tag{3}$$

After convergence of $G$, this will be equivalent to equation 2, and will converge to the same attractor. The stable points of 3 occur when $E[uz^T - Nzz^T] = 0$, for which $N = R^T S^{-1}$

•

The convergence behavior of the Double Generalized Hebbian Algorithm is shown in figure 4. Results are measured by computing $B = GPN$ and determining whether $B$ is diagonal using a score

$$\epsilon = \frac{\sum_{i \neq j} b_{ij}^2}{\sum_i b_i^2}$$

The reduction in $\epsilon$ is shown as a function of the number of $(u, y)$ examples given to the network during training, and the curves in the figure represent the average over 100 training runs with different randomly-selected plant matrices $P$.

Note that the Double Generalized Hebbian Algorithm may perform poorly in the presence of noise or uncontrollable modes. The sensory mapping $G$ depends only on the outputs $y$, and not directly on the plant inputs $u$. So if the outputs include noise or autonomously varying uncontrollable modes, then the mapping $G$ will respond to these modes. This is not a problem if most of the variance in the output is due the inputs $u$, since in that case the most significant output components will reflect the input variance transmitted through $P$.

## 4    THE ORTHOGONAL ASYMMETRIC ENCODER

The second algorithm is the Orthogonal Asymmetric Encoder (OAE) which is described by the equations

$$\Delta G \quad = \quad \gamma(\hat{z}y^T - LT[\hat{z}\hat{z}^T]G)\tag{4}$$

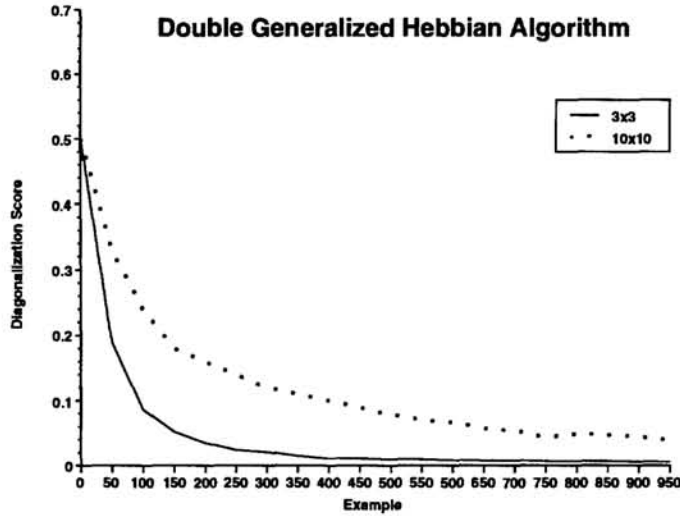

Figure 4: Convergence of the Double Generalized Hebbian Algorithm averaged over 100 random choices of 3x3 or 10x10 matrices $P$.

$$\Delta N^T \;=\; \gamma(Gy - \mathrm{LT}[GG^T]\hat{z})u^T \tag{5}$$

where $\hat{z} = N^T u$.

This algorithm uses a variant of the Backpropagation learning algorithm (Rumelhart *et al.* 1986). It is named for the "Encoder" problem in which a three-layer network is trained to approximate the identity mapping but is forced to use a narrow bottleneck layer. I define the "Asymmetric Encoder Problem" as the case in which a mapping other than the identity is to be learned while the data is passed through a bottleneck. The "Orthogonal Asymmetric Encoder" (OAE) is the special case in which the hidden units are forced to be uncorrelated over the data set. Figure 5 gives a graphical depiction of the algorithm.

**Theorem 2:** *(Sanger 1993) Equations 4 and 5 converge to the left and right singular vectors of $P$.*

Proof:

Suppose $z$ has dimension $m$. If $P = L^T S R$ where the elements of $S$ are distinct, and $E[uu^T] = I$, then a well-known property of the singular value decomposition (Golub and Van Loan 1983, , for example) shows that

$$E[\|Pu - G^T N^T u\|] \tag{6}$$

is minimized when $G^T = L_m^T U$, $N^T = V R_m$, and $U$ and $V$ are any $m \times m$ matrices for which $UV = I_m S I_m^T$. ($L_m^T$ and $R_m$ signify the matrices of only the first $m$ columns of $L^T$ or rows of $R$.) If we want $E[zz^T]$ to be diagonal, then $U$ and $V$ must be diagonal. OAE accomplishes this by training the first hidden unit as if $m = 1$, the second as if $m = 2$, and so on.

For the case $m = 1$, the error 6 is minimized when $G$ is the first left singular vector of $P$ and $N$ is the first right singular vector. Since this is a linear approximation problem, there is a single global minimum to the error surface 6, and gradient descent using the backpropagation algorithm will converge to this solution.

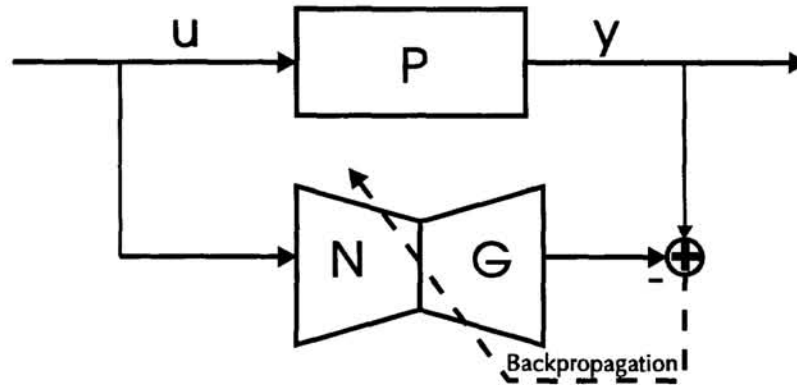

Figure 5: The Orthogonal Asymmetric Encoder algorithm computes a forward approximation to the plant $P$ through a bottleneck layer of hidden units.

After convergence, the remaining error is $E[\||(P - G^T N^T)u\||]$. If we decompose the plant matrix as

$$P = \sum_{i=1}^{n} l_i s_i r_i^T$$

where $l_i$ and $r_i$ are the rows of $L$ and $R$, and $s_i$ are the diagonal elements of $S$, then the remaining error is

$$P_2 = \sum_{i=2}^{n} l_i s_i r_i^T$$

which is equivalent to the original plant matrix with the first singular value set to zero. If we train the second hidden unit using $P_2$ instead of $P$, then minimization of $E[\||P_2 u - G^T N^T u\||]$ will yield the second left and right singular vectors. Proceeding in this way we can obtain the first $m$ singular vectors.

Combining the update rules for all the singular vectors so that they learn in parallel leads to the governing equations of the OAE algorithm which can be written in matrix form as equations 4 and 5.

●

(Bannour and Azimi-Sadjadi 1993) proposed a similar technique for the symmetric encoder problem in which each eigenvector is learned to convergence and then subtracted from the data before learning the succeeding one. The orthogonal asymmetric encoder is different because all the components learn simultaneously. After convergence, we must multiply the learned $N$ by $S^{-2}$ in order to compute the plant inverse. Figure 6 shows the performance of the algorithm averaged over 100 random choices of matrix $P$.

Consider the case in which there may be noise in the measured outputs $y$. Since the Orthogonal Asymmetric Encoder algorithm learns to approximate the forward plant transformation from $u$ to $y$, it will only be able to predict the components of $y$ which are related to the inputs $u$. In other words, the best approximation to $y$ based on $u$ is $\hat{y} \approx Pu$, and this ignores the noise term. Figure 7 shows the results of additive noise with an SNR of 1.0.

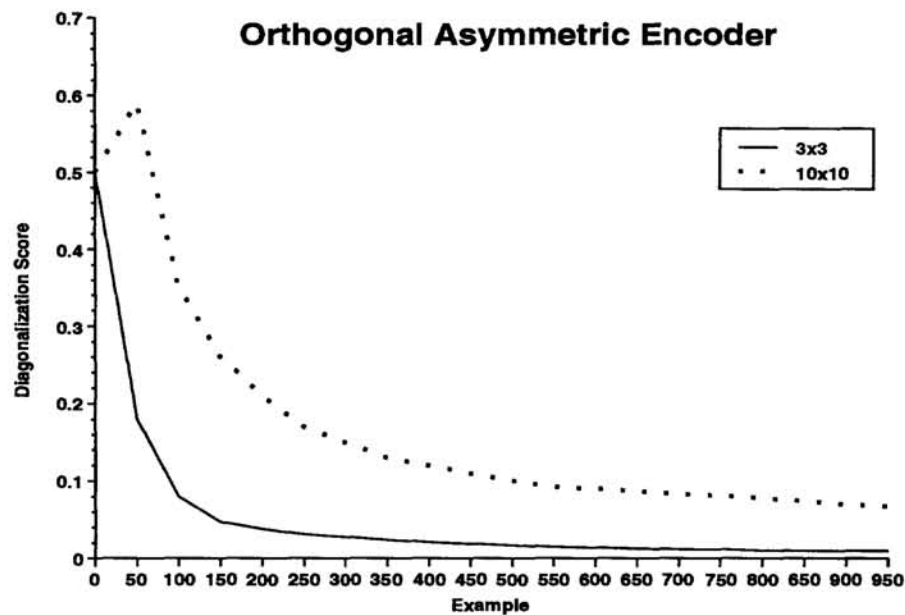

Figure 6: Convergence of the Orthogonal Asymmetric Encoder averaged over 100 random choices of 3x3 or 10x10 matrices $P$.

## Acknowledgements

This report describes research done within the laboratory of Dr. Emilio Bizzi in the department of Brain and Cognitive Sciences at MIT. The author was supported during this work by a National Defense Science and Engineering Graduate Fellowship, and by NIH grants 5R37AR26710 and 5R01NS09343 to Dr. Bizzi.

## References

Bannour S., Azimi-Sadjadi M. R., 1993, Principal component extraction using recursive least squares learning, submitted to *IEEE Transactions on Neural Networks*.

Golub G. H., Van Loan C. F., 1983, *Matrix Computations*, North Oxford Academic P., Oxford.

Rumelhart D. E., Hinton G. E., Williams R. J., 1986, Learning internal representations by error propagation, In *Parallel Distributed Processing*, chapter 8, pages 318–362, MIT Press, Cambridge, MA.

Sanger T. D., 1989, Optimal unsupervised learning in a single-layer linear feedforward neural network, *Neural Networks*, 2:459–473.

Sanger T. D., 1993, *Theoretical Elements of Hierarchical Control in Vertebrate Motor Systems*, PhD thesis, MIT.

Widrow B., Hoff M. E., 1960, Adaptive switching circuits, In *IRE WESCON Conv. Record, Part 4*, pages 96–104.

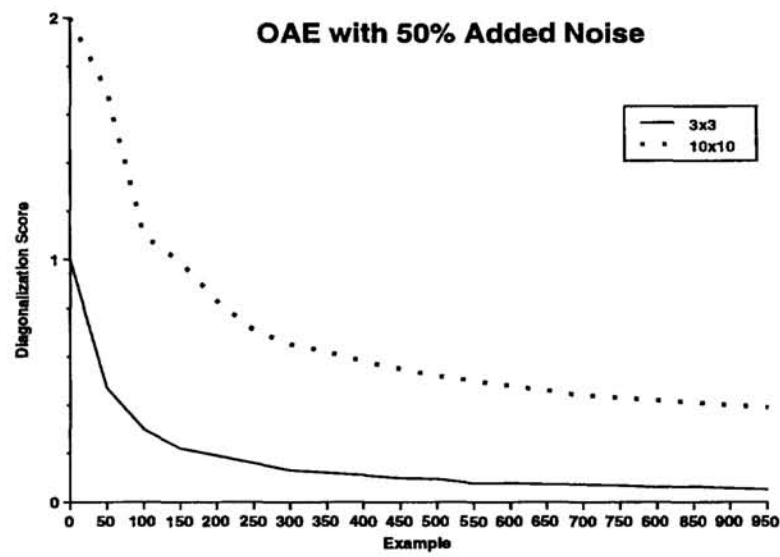

Figure 7: Convergence of the Orthogonal Asymmetric Encoder with 50% additive noise on the outputs, averaged over 100 random choices of 3x3 or 10x10 matrices $P$.